# Constrained Differential Optimization

John C. Platt
Alan H. Barr

California Institute of Technology, Pasadena, CA 91125

## Abstract

Many optimization models of neural networks need constraints to restrict the space of outputs to a subspace which satisfies external criteria. Optimizations using energy methods yield "forces" which act upon the state of the neural network. The penalty method, in which quadratic energy constraints are added to an existing optimization energy, has become popular recently, but is not guaranteed to satisfy the constraint conditions when there are other forces on the neural model or when there are multiple constraints. In this paper, we present the *basic differential multiplier method* (BDMM), which satisfies constraints exactly; we create forces which gradually apply the constraints over time, using "neurons" that estimate Lagrange multipliers.

The basic differential multiplier method is a differential version of the method of multipliers from Numerical Analysis. We prove that the differential equations locally converge to a constrained minimum.

Examples of applications of the differential method of multipliers include enforcing permutation codewords in the analog decoding problem and enforcing valid tours in the traveling salesman problem.

## 1. Introduction

Optimization is ubiquitous in the field of neural networks. Many learning algorithms, such as back-propagation,[18] optimize by minimizing the difference between expected solutions and observed solutions. Other neural algorithms use differential equations which minimize an energy to solve a specified computational problem, such as associative memory,[9] differential solution of the traveling salesman problem,[5,10] analog decoding,[15] and linear programming.[19] Furthermore, Lyapunov methods show that various models of neural behavior find minima of particular functions.[4,9]

Solutions to a constrained optimization problem are restricted to a subset of the solutions of the corresponding unconstrained optimization problem. For example, a mutual inhibition circuit[6] requires one neuron to be "on" and the rest to be "off". Another example is the traveling salesman problem,[13] where a salesman tries to minimize his travel distance, subject to the constraint that he must visit every city exactly once. A third example is the curve fitting problem, where elastic splines are as smooth as possible, while still going through data points.[3] Finally, when digital decisions are being made on analog data, the answer is constrained to be bits, either 0 or 1.[14]

A constrained optimization problem can be stated as

$$\text{minimize } f(\underline{x}),$$
$$\text{subject to } g(\underline{x}) = 0, \tag{1}$$

where $\underline{x}$ is the state of the neural network, a position vector in a high-dimensional space; $f(\underline{x})$ is a scalar energy, which can be imagined as the height of a landscape as a function of position $\underline{x}$; $g(\underline{x}) = 0$ is a scalar equation describing a subspace of the state space. During constrained optimization, the state should be attracted to the subspace $g(\underline{x}) = 0$, then slide along the subspace until it reaches the locally smallest value of $f(\underline{x})$ on $g(\underline{x}) = 0$.

In section 2 of the paper, we describe classical methods of constrained optimization, such as the penalty method and Lagrange multipliers.

Section 3 introduces the basic differential multiplier method (BDMM) for constrained optimization, which calculates a good local minimum. If the constrained optimization problem is convex, then the local minimum is the global minimum; in general, finding the global minimum of non-convex problems is fairly difficult.

In section 4, we show a Lyapunov function for the BDMM by drawing on an analogy from physics.

In section 5, augmented Lagrangians, an idea from optimization theory, enhances the convergence properties of the BDMM.

In section 6, we apply the differential algorithm to two neural problems, and discuss the insensitivity of BDMM to choice of parameters. Parameter sensitivity is a persistent problem in neural networks.

## 2. Classical Methods of Constrained Optimization

This section discusses two methods of constrained optimization, the penalty method and Lagrange multipliers. The penalty method has been previously used in differential optimization. The basic differential multiplier method developed in this paper applies Lagrange multipliers to differential optimization.

### 2.1. The Penalty Method

The penalty method is analogous to adding a rubber band which attracts the neural state to the subspace $g(\underline{x}) = 0$. The penalty method adds a quadratic energy term which penalizes violations of constraints. [8] Thus, the constrained minimization problem (1) is converted to the following unconstrained minimization problem:

$$\text{minimize } \mathcal{E}_{\text{penalty}}(\underline{x}) = f(\underline{x}) + c(g(\underline{x}))^2. \tag{2}$$

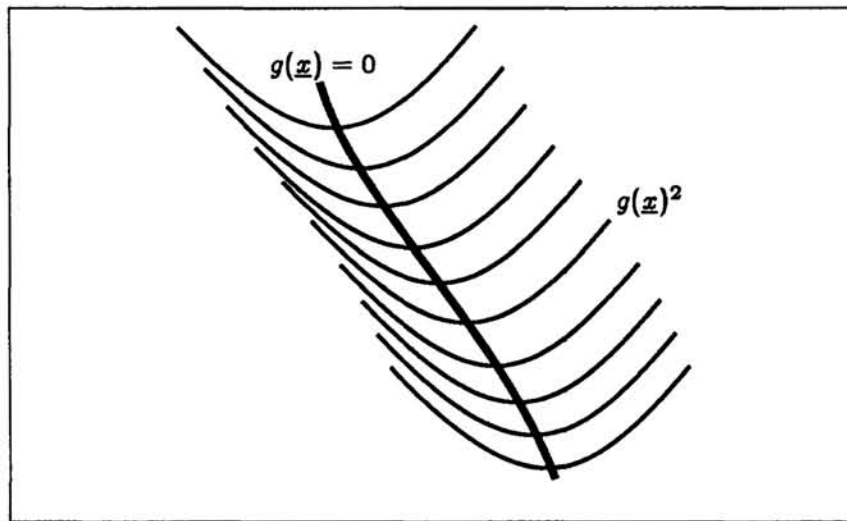

**Figure 1.** The penalty method makes a trough in state space

The penalty method can be extended to fulfill multiple constraints by using more than one rubber band. Namely, the constrained optimization problem

$$\begin{aligned} &\text{minimize } f(\underline{x}), \\ &\text{subject to } g_\alpha(\underline{x}) = 0; \quad \alpha = 1, 2, \dots, n; \end{aligned} \tag{3}$$

is converted into unconstrained optimization problem

$$\text{minimize } \mathcal{E}_{\text{penalty}}(\underline{x}) = f(\underline{x}) + \sum_{\alpha=1}^{n} c_\alpha (g_\alpha(\underline{x}))^2. \tag{4}$$

The penalty method has several convenient features. First, it is easy to use. Second, it is globally convergent to the correct answer as $c_\alpha \to \infty$.[8] Third, it allows compromises between constraints. For example, in the case of a spline curve fitting input data, there can be a compromise between fitting the data and making a smooth spline.

However, the penalty method has a number of disadvantages. First, for finite constraint strengths $c_\alpha$, it doesn't fulfill the constraints exactly. Using multiple rubber band constraints is like building a machine out of rubber bands: the machine would not hold together perfectly. Second, as more constraints are added, the constraint strengths get harder to set, especially when the size of the network (the dimensionality of $\underline{x}$) gets large.

In addition, there is a dilemma to the setting of the constraint strengths. If the strengths are small, then the system finds a deep local minimum, but does not fulfill all the constraints. If the strengths are large, then the system quickly fulfills the constraints, but gets stuck in a poor local minimum.

### 2.2. Lagrange Multipliers

Lagrange multiplier methods also convert constrained optimization problems into unconstrained extremization problems. Namely, a solution to the equation (1) is also a critical point of the energy

$$\mathcal{E}_{\text{Lagrange}}(\underline{x}) = f(\underline{x}) + \lambda g(\underline{x}). \tag{5}$$

$\lambda$ is called the Lagrange multiplier for the constraint $g(\underline{x}) = 0$.[8]

A direct consequence of equation (5) is that the gradient of $f$ is collinear to the gradient of $g$ at the constrained extrema (see Figure 2). The constant of proportionality between $\nabla f$ and $\nabla g$ is $-\lambda$:

$$\nabla \mathcal{E}_{\text{Lagrange}} = 0 = \nabla f + \lambda \nabla g. \tag{6}$$

We use the collinearity of $\nabla f$ and $\nabla g$ in the design of the BDMM.

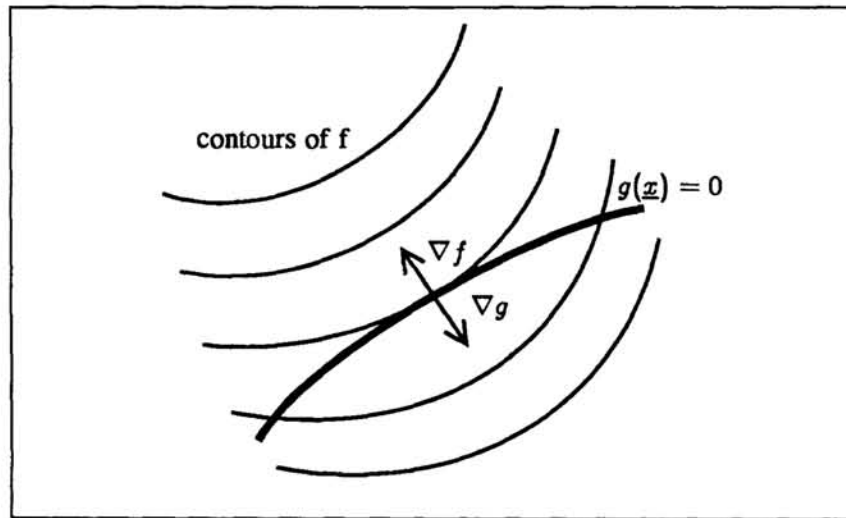

**Figure 2.** At the constrained minimum, $\nabla f = -\lambda \nabla g$

A simple example shows that Lagrange multipliers provide the extra degrees of freedom necessary to solve constrained optimization problems. Consider the problem of finding a point $(x, y)$ on the line $x + y = 1$ that is closest to the origin. Using Lagrange multipliers,

$$\mathcal{E}_{\text{Lagrange}} = x^2 + y^2 + \lambda(x + y - 1) \tag{7}$$

Now, take the derivative with respect to all variables, $x, y$, and $\lambda$.

$$\frac{\partial \mathcal{E}_{\text{Lagrange}}}{\partial x} = 2x + \lambda = 0$$

$$\frac{\partial \mathcal{E}_{\text{Lagrange}}}{\partial y} = 2y + \lambda = 0 \tag{8}$$

$$\frac{\partial \mathcal{E}_{\text{Lagrange}}}{\partial \lambda} = x + y - 1 = 0$$

With the extra variable $\lambda$, there are now three equations in three unknowns. In addition, the last equation is precisely the constraint equation.

## 3. The Basic Differential Multiplier Method for Constrained Optimization

This section presents a new "neural" algorithm for constrained optimization, consisting of differential equations which estimate Lagrange multipliers. The neural algorithm is a variation of the method of multipliers, first presented by Hestenes[9] and Powell[16].

### 3.1. Gradient Descent does not work with Lagrange Multipliers

The simplest differential optimization algorithm is *gradient descent*, where the state variables of the network slide downhill, opposite the gradient. Applying gradient descent to the energy in equation (5) yields

$$\dot{x}_i = -\frac{\partial \mathcal{E}_{\text{Lagrange}}}{\partial x_i} = -\frac{\partial f}{\partial x_i} - \lambda \frac{\partial g}{\partial x_i},$$

$$\dot{\lambda} = -\frac{\partial \mathcal{E}_{\text{Lagrange}}}{\partial \lambda} = -g(\underline{x}). \tag{9}$$

Note that there is a auxiliary differential equation for $\lambda$, which is an additional "neuron" necessary to apply the constraint $g(\underline{x}) = 0$. Also, recall that when the system is at a constrained extremum, $\nabla f = -\lambda \nabla g$, hence, $\dot{x}_i = 0$.

Energies involving Lagrange multipliers, however, have critical points which tend to be saddle points. Consider the energy in equation (5). If $\underline{x}$ is frozen, the energy can be decreased by sending $\lambda$ to $+\infty$ or $-\infty$.

Gradient descent does not work with Lagrange multipliers, because a critical point of the energy in equation (5) need not be an attractor for (9). A stationary point must be a local minimum in order for gradient descent to converge.

### 3.2. The New Algorithm: the Basic Differential Multiplier Method

We present an alternative to differential gradient descent that estimates the Lagrange multipliers, so that the constrained minima are attractors of the differential equations, instead of "repulsors." The differential equations that solve (1) is

$$\dot{x}_i = -\frac{\partial f}{\partial x_i} - \lambda \frac{\partial g}{\partial x_i},$$

$$\dot{\lambda} = +g(\underline{x}). \tag{10}$$

Equation (10) is similar to equation (9). As in equation (9), constrained extrema of the energy (5) are stationary points of equation (10). Notice, however, the sign inversion in the equation for $\lambda$, as compared to equation (9). The equation (10) is performing gradient *ascent* on $\lambda$. The sign flip makes the BDMM stable, as shown in section 4.

Equation (10) corresponds to a neural network with anti-symmetric connections between the $\lambda$ neuron and all of the $\underline{x}$ neurons.

### 3.3. Extensions to the Algorithm

One extension to equation (10) is an algorithm for constrained minimization with multiple constraints. Adding an extra neuron for every equality constraint and summing all of the constraint forces creates the energy

$$\mathcal{E}_{\text{multiple}} = f(\underline{x}) + \sum_{\alpha} \lambda_{\alpha} g_{\alpha}(\underline{x}), \tag{11}$$

which yields differential equations

$$\dot{x}_i = -\frac{\partial f}{\partial x_i} - \sum_{\alpha} \lambda_{\alpha} \frac{\partial g\alpha}{\partial x_i},$$

$$\dot{\lambda}_{\alpha} = +g_{\alpha}(\underline{x}). \tag{12}$$

Another extension is constrained minimization with inequality constraints. As in traditional optimization theory,[8] one uses extra slack variables to convert inequality constraints into equality constraints. Namely, a constraint of the form $h(\underline{x}) \geq 0$ can be expressed as

$$g(\underline{x}) = h(\underline{x}) - z^2. \tag{13}$$

Since $z^2$ must always be positive, then $h(\underline{x})$ is constrained to be positive. The slack variable $z$ is treated like a component of $\underline{x}$ in equation (10). An inequality constraint requires two extra neurons, one for the slack variable $x$ and one for the Lagrange multiplier $\lambda$.

Alternatively, the inequality constraint can be represented as an equality constraint. For example, if $h(\underline{x}) \geq 0$, then the optimization can be constrained with $g(\underline{x}) = h(\underline{x})$, when $h(\underline{x}) \geq 0$; and $g(\underline{x}) = 0$ otherwise.

## 4. Why the algorithm works

The system of differential equations (10) (the BDMM) gradually fulfills the constraints. Notice that the function $g(\underline{x})$ can be replaced by $kg(\underline{x})$, without changing the location of the constrained minimum. As $k$ is increased, the state begins to undergo damped oscillation about the constraint subspace $g(\underline{x}) = 0$. As $k$ is increased further, the frequency of the oscillations increase, and the time to convergence increases.

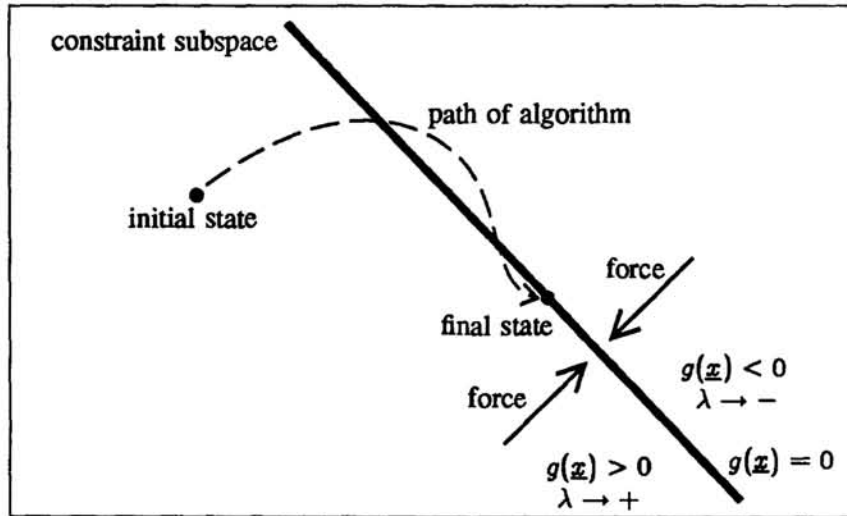

**Figure 3.** The state is attracted to the constraint subspace

The damped oscillations of equation (10) can be explained by combining both of the differential equations into one second-order differential equation.

$$\ddot{x}_i + \sum_j \left( \frac{\partial^2 f}{\partial x_i \partial x_j} + \lambda \frac{\partial^2 g}{\partial x_i \partial x_j} \right) \dot{x}_j + g \frac{\partial g}{\partial x_i} = 0. \tag{14}$$

Equation (14) is the equation for a damped mass system, with an inertia term $\ddot{x}_i$, a damping matrix

$$A_{ij} = \frac{\partial^2 f}{\partial x_i \partial x_j} + \lambda \frac{\partial^2 g}{\partial x_i \partial x_j}, \tag{15}$$

and an internal force, $g \partial g / \partial x_i$, which is the derivative of the internal energy

$$U = \frac{1}{2}(g(\underline{x}))^2. \tag{16}$$

If the system is damped and the state remains bounded, the state falls into a constrained minima.

As in physics, we can construct a total energy of the system, which is the sum of the kinetic and potential energies.

$$E = T + U = \sum_i \frac{1}{2}(\dot{x}_i)^2 + \frac{1}{2}(g(\underline{x}))^2. \tag{17}$$

If the total energy is decreasing with time and the state remains bounded, then the system will dissipate any extra energy, and will settle down into the state where

$$g(\underline{x}) = 0,$$
$$\dot{x}_i = \frac{\partial f}{\partial x_i} + \lambda \frac{\partial g}{\partial x_i} = 0. \tag{18}$$

which is a constrained extremum of the original problem in equation (1).

The time derivative of the total energy in equation (17) is

$$\begin{aligned} \dot{E} &= \sum_i \ddot{x}_i \dot{x}_i + g(\underline{x}) \frac{\partial g}{\partial x_i} \dot{x}_i \\ &= -\sum_{i,j} \dot{x}_i A_{ij} \dot{x}_j. \end{aligned} \tag{19}$$

If damping matrix $A_{ij}$ is positive definite, the system converges to fulfill the constraints.

BDMM always converges for a special case of constrained optimization: quadratic programming. A quadratic programming problem has a quadratic function $f(\underline{x})$ and a piecewise linear continuous function $g(\underline{x})$ such that

$$\frac{\partial^2 f}{\partial x_i \partial x_j} \text{ is positive definite;} \qquad \frac{\partial^2 g}{\partial x_i \partial x_j} = 0. \tag{20}$$

Under these circumstances, the damping matrix $A_{ij}$ is positive definite for all $\underline{x}$ and $\lambda$, so that the system converges to the constraints.

### 4.1. Multiple constraints

For the case of multiple constraints, the total energy for equation (12) is

$$E = T + U = \sum_i \frac{1}{2}(\dot{x}_i)^2 + \sum_\alpha \frac{1}{2}g_\alpha(\underline{x})^2. \tag{21}$$

and the time derivative is

$$\begin{aligned} \dot{E} &= \sum_i \ddot{x}_i \dot{x}_i + \sum_\alpha g_\alpha(\underline{x}) \frac{\partial g_\alpha}{\partial x_i} \dot{x}_i \\ &= -\sum_{i,j} \dot{x}_i \left( \frac{\partial^2 f}{\partial x_i \partial x_j} + \sum_\alpha \lambda_\alpha \frac{\partial^2 g_\alpha}{\partial x_i \partial x_j} \right) \dot{x}_j. \end{aligned} \tag{22}$$

Again, BDMM solves a quadratic programming problem, if a solution exists. However, it is possible to pose a problem that has contradictory constraints. For example,

$$g_1(x) = x = 0, \qquad g_2(x) = x - 1 = 0 \tag{23}$$

In the case of conflicting constraints, the BDMM compromises, trying to make each constraint $g_\alpha$ as small as possible. However, the Lagrange multipliers $\lambda_\alpha$ goes to $\pm\infty$ as the constraints oppose each other. It is possible, however, to arbitrarily limit the $\lambda_\alpha$ at some large absolute value.

LaSalle's invariance theorem[12] is used to prove that the BDMM eventually fulfills the constraints. Let $G$ be an open subset of $R^n$. Let $F$ be a subset of $G^*$, the closure of $G$, where the system of differential equations (12) is at an equilibrium.

$$F = \{\underline{x}, \underline{\lambda} \mid \dot{x}_i = 0, \dot{\lambda}_\alpha = 0, \underline{x}, \underline{\lambda} \in G^*\} \tag{24}$$

If the damping matrix

$$\frac{\partial^2 f}{\partial x_i \partial x_j} + \sum_\alpha \lambda_\alpha \frac{\partial^2 g_\alpha}{\partial x_i \partial x_j} \tag{25}$$

is positive definite in G, if $x_i(t)$ and $\lambda_\alpha(t)$ are bounded, and remain in $G$ for all time, and if $F$ is non-empty, then $F$ is the largest invariant set in $G^*$, hence, by LaSalle's invariance theorem, the system $x_i(t), \lambda_\alpha(t)$ approaches $F$ as $t \to \infty$.

## 5. The Modified Differential Method of Multipliers

This section presents the *modified differential multiplier method* (MDMM), which is a modification of the BDMM with more robust convergence properties. For a given constrained optimization problem, it is frequently necessary to alter the BDMM to have a region of positive damping surrounding the constrained minima. The non-differential method of multipliers from Numerical Analysis also has this difficulty. [2] Numerical Analysis combines the multiplier method with the penalty method to yield a modified multiplier method that is locally convergent around constrained minima. [2]

The BDMM is completely compatible with the penalty method. If one adds a penalty force to equation (10) corresponding to an quadratic energy

$$E_{\text{penalty}} = \frac{c}{2}(g(\underline{x}))^2. \tag{26}$$

then the set of differential equations for MDMM is

$$\dot{x}_i = -\frac{\partial f}{\partial x_i} - \lambda \frac{\partial g}{\partial x_i} - cg\frac{\partial g}{\partial x_i}, \tag{27}$$
$$\dot{\lambda} = g(\underline{x}).$$

The extra force from the penalty does *not* change the position of the stationary points of the differential equations, because the penalty force is 0 when $g(\underline{x}) = 0$. The damping matrix is modified by the penalty force to be

$$A_{ij} = \frac{\partial^2 f}{\partial x_i \partial x_j} + \lambda \frac{\partial^2 g}{\partial x_i \partial x_j} + c\frac{\partial g}{\partial x_i}\frac{\partial g}{\partial x_j} + cg\frac{\partial^2 g}{\partial x_i \partial x_j}. \tag{28}$$

There is a theorem [1] that states that there exists a $c^* > 0$ such that if $c > c^*$, the damping matrix in equation (28) is positive definite at constrained minima. Using continuity, the damping matrix is positive definite in a region $R$ surrounding each constrained minimum. If the system starts in the region $R$ and remains bounded and in $R$, then the convergence theorem at the end of section 4 is applicable, and MDMM will converge to a constrained minimum.

The minimum necessary penalty strength $c$ for the MDMM is usually much less than the strength needed by the penalty method alone.[2]

## 6. Examples

This section contains two examples which illustrate the use of the BDMM and the MDMM. First, the BDMM is used to find a good solution to the planar traveling salesman problem. Second, the MDMM is used to enforcing mutual inhibition and digital results in the task of analog decoding.

### 6.1. Planar Traveling Salesman

The traveling salesman problem (TSP) is, given a set of cities lying in the plane, find the shortest closed path that goes through every city exactly once. Finding the shortest path is NP-complete.

Finding a nearly optimal path, however, is much easier than finding a globally optimal path. There exist many heuristic algorithms for approximately solving the traveling salesman problem.[5,10,11,13] The solution presented in this section is moderately effective and illustrates the independence of BDMM to changes in parameters.

Following Durbin and Willshaw,[5] we use an elastic snake to solve the TSP. A snake is a discretized curve which lies on the plane. The elements of the snake are points on the plane, $(x_i, y_i)$. A snake is a locally connected neural network, whose neural outputs are positions on the plane.

The snake minimizes its length

$$\sum_i (x_{i+1} - x_i)^2 - (y_{i+1} - y_i)^2,$$ 
$$(29)$$

subject to the constraint that the snake must lie on the cities:

$$k(x^* - x_c) = 0, \qquad k(y^* - y_c) = 0,$$ 
$$(30)$$

where $(x^*, y^*)$ are city coordinates, $(x_c, y_c)$ is the closest snake point to the city, and $k$ is the constraint strength.

The minimization in equation (29) is quadratic and the constraints in equation (30) are piecewise linear, corresponding to a $C^0$ continuous potential energy in equation (21). Thus, the damping is positive definite, and the system converges to a state where the constraints are fulfilled.

In practice, the snake starts out as a circle. Groups of cities grab onto the snake, deforming it. As the snake gets close to groups of cities, it grabs onto a specific ordering of cities that locally minimize its length (see Figure 4).

The system of differential equations that solve equations (29) and (30) are piecewise linear. The differential equations for $x_i$ and $y_i$ are solved with implicit Euler's method, using tridiagonal LU decomposition to solve the linear system.[17] The points of the snake are sorted into bins that divide the plane, so that the computation of finding the nearest point is simplified.

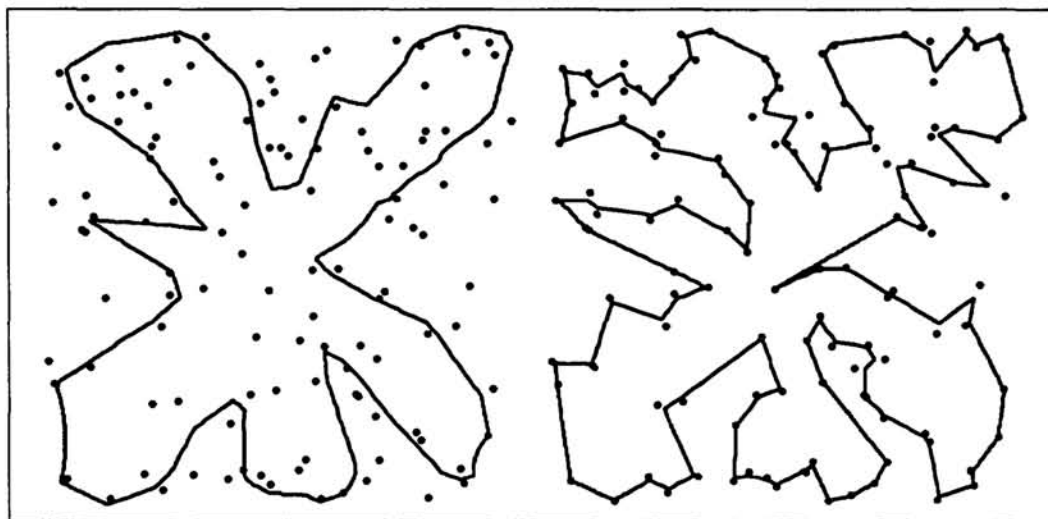

**Figure 4.** The snake eventually attaches to the cities

The constrained minimization in equations (29) and (30) is a reasonable method for approximately solving the TSP. For 120 cities distributed in the unti square, and 600 snake points, a numerical step size of 100 time units, and a constraint strength of $5 \times 10^{-3}$, the tour lengths are $6\% \pm 2\%$ longer than that yielded by simulated annealing[11]. Empirically, for 30 to 240 cities, the time needed to compute the final city ordering scales as $N^{1.6}$, as compared to the Kernighan-Lin method[13], which scales roughly as $N^{2.2}$.

The constraint strength is usable for both a 30 city problem and a 240 city problem. Although changing the constraint strength affects the performance, the snake attaches to the cities for any non-zero constraint strength. Parameter adjustment does not seem to be an issue as the number of cities increases, unlike the penalty method.

## 6.2. Analog Decoding

Analog decoding uses analog signals from a noisy channel to reconstruct codewords. Analog decoding has been performed neurally,[15] with a code space of permutation matrices, out of the possible space of binary matrices.

To perform the decoding of permutation matrices, the nearest permutation matrix to the signal matrix must be found. In other words, find the nearest matrix to the signal matrix, subject to the constraint that the matrix has on/off binary elements, and has exactly one "on" per row and one "on" per column. If the signal matrix is $I_{ij}$ and the result is $V_{ij}$, then minimize

$$-\sum_{i,j} V_{ij} I_{ij} \tag{31}$$

subject to constraints

$$V_{ij}(1 - V_{ij}) = 0; \qquad \sum_i V_{ij} - 1 = 0; \qquad \sum_j V_{ij} - 1 = 0. \tag{32}$$

In this example, the first constraint in equation (32) forces crisp digital decisions. The second and third constraints are mutual inhibition along the rows and columns of the matrix.

The optimization in equation (31) is not quadratic, it is linear. In addition, the first constraint in equation (32) is non-linear. Using the BDMM results in undamped oscillations. In order to converge onto a constrained minimum, the MDMM must be used. For both a $5 \times 5$ and a $20 \times 20$ system, a $c = 0.2$ is adequate for damping the oscillations. The choice of $c$ seems to be reasonably insensitive to the size of the system, and a wide range of $c$, from 0.02 to 2.0, damps the oscillations.

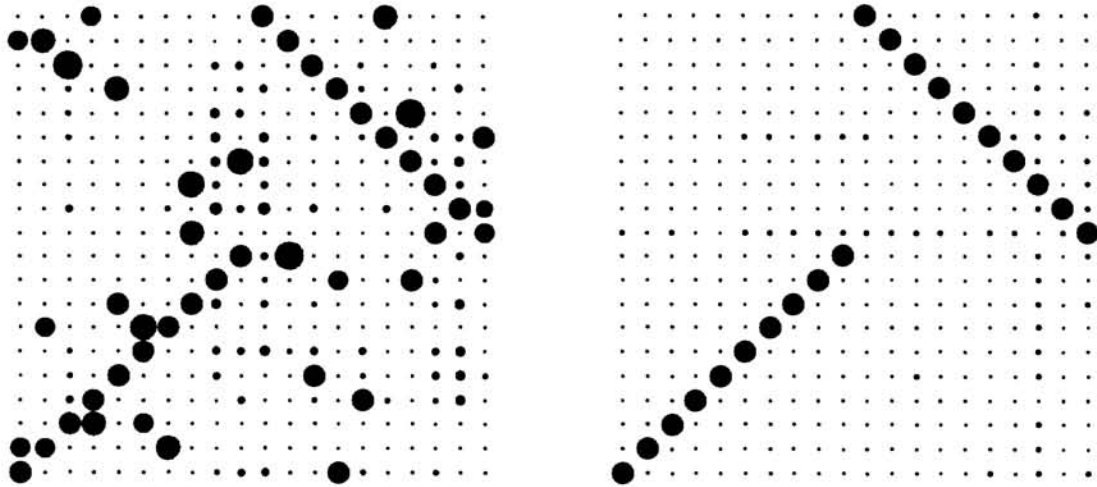

**Figure 5.** The decoder finds the nearest permutation matrix

In a test of the MDMM, a signal matrix which is a permutation matrix plus some noise, with a signal-to-noise ratio of 4 is supplied to the network. In figure 5, the system has turned on the correct neurons but also many incorrect neurons. The constraints start to be applied, and eventually the system reaches a permutation matrix. The differential equations do not need to be reset. If a new signal matrix is applied to the network, the neural state will move towards the new solution.

## 7. Conclusions

In the field of neural networks, there are differential optimization algorithms which find local solutions to non-convex problems. The basic differential multiplier method is a modification of a standard constrained optimization algorithm, which improves the capability of neural networks to perform constrained optimization.

The BDMM and the MDMM offer many advantages over the penalty method. First, the differential equations (10) are much less stiff than those of the penalty method. Very large quadratic terms are not needed by the MDMM in order to strongly enforce the constraints. The energy terrain for the

penalty method looks like steep canyons, with gentle floors; finding minima of these types of energy surfaces is numerically difficult. In addition, the steepness of the penalty terms is usually sensitive to the dimensionality of the space. The differential multiplier methods are promising techniques for alleviating stiffness.

The differential multiplier methods separate the speed of fulfilling the constraints from the accuracy of fulfilling the constraints. In the penalty method, as the strengths of a constraint goes to $\infty$, the constraint is fulfilled, but the energy has many undesirable local minima. The differential multiplier methods allow one to choose how quickly to fulfill the constraints.

The BDMM fulfills constraints exactly and is compatible with the penalty method. Addition of penalty terms in the MDMM does not change the stationary points of the algorithm, and sometimes helps to damp oscillations and improve convergence.

Since the BDMM and the MDMM are in the form of first-order differential equations, they can be directly implemented in hardware. Performing constrained optimization at the raw speed of analog VLSI seems like a promising technique for solving difficult perception problems.[14]

There exist Lyapunov functions for the BDMM and the MDMM. The BDMM converges globally for quadratic programming. The MDMM is provably convergent in a local region around the constrained minima. Other optimization algorithms, such as Newton's method,[17] have similar local convergence properties. The global convergence properties of the BDMM and the MDMM are currently under investigation.

In summary, the differential method of multipliers is a useful way of enforcing constraints on neural networks for enforcing syntax of solutions, encouraging desirable properties of solutions, and making crisp decisions.

## Acknowledgments

This paper was supported by an AT&T Bell Laboratories Fellowship (JCP).

## References

1. K. J. Arrow, L. Hurwicz, H. Uzawa, *Studies in Linear and Nonlinear Programming*, (Stanford University Press, Stanford, CA, 1958).
2. D. P. Bertsekas, *Automatica*, **12**, 133-145, (1976).
3. C. de Boor, *A Practical Guide to Splines*, (Springer-Verlag, NY, 1978).
4. M. A. Cohen, S. Grossberg, *IEEE Trans. Systems, Man, and Cybernetics*, , 815-826, (1983).
5. R. Durbin, D. Willshaw, *Nature*, **326**, 689-691, (1987).
6. J. C. Eccles, *The Physiology of Nerve Cells*, (Johns Hopkins Press, Baltimore, 1957).
7. M. R. Hestenes, *J. Opt. Theory Appl.*, **4**, 303-320, (1969).
8. M. R. Hestenes, *Optimization Theory*, (Wiley & Sons, NY, 1975).
9. J. J. Hopfield, *PNAS*, **81**, 3088, (1984).
10. J. J. Hopfield, D. W. Tank, *Biological Cybernetics*, **52**, 141, (1985).
11. S. Kirkpatrick, C. D. Gelatt, C. M. Vecchi, *Science*, **220**, 671-680, (1983).
12. J. LaSalle, *The Stability of Dynamical Systems*, (SIAM, Philadelphia, 1976).
13. S. Lin, B. W. Kernighan, *Oper. Res.*, **21**, 498-516 (1973).
14. C. A. Mead, *Analog VLSI and Neural Systems*, (Addison-Wesley, Reading, MA, TBA).
15. J. C. Platt, J. J. Hopfield, in *AIP Conf. Proc. 151: Neural Networks for Computing* (J. Denker ed.) 364-369, (American Institute of Physics, NY, 1986).
16. M. J. Powell, in *Optimization*, (R. Fletcher, ed.), 283-298, (Academic Press, NY, 1969).
17. W. H. Press, B. P. Flannery, S. A. Teukolsky, W. T. Vetterling, *Numerical Recipes*, (Cambridge University Press, Cambridge, 1986).
18. D. Rumelhart, G. Hinton, R. Williams, in *Parallel Distributed Processing*, (D. Rumelhart, ed.), **1**, 318-362, (MIT Press, Cambridge, MA, 1986).
19. D. W. Tank, J. J. Hopfield, *IEEE Trans. Cir. & Sys.*, **CAS-33**, no. 5, 533-541 (1986).
